# PICODES: Learning a Compact Code for Novel-Category Recognition

**Alessandro Bergamo, Lorenzo Torresani**
Dartmouth College
Hanover, NH, U.S.A.
{aleb, lorenzo}@cs.dartmouth.edu

**Andrew Fitzgibbon**
Microsoft Research
Cambridge, United Kingdom
awf@microsoft.com

## Abstract

We introduce PICODES: a very compact image descriptor which nevertheless allows high performance on object category recognition. In particular, we address novel-category recognition: the task of defining indexing structures and image representations which enable a large collection of images to be searched for an object category that was not known when the index was built. Instead, the training images defining the category are supplied at query time. We explicitly learn descriptors of a given length (from as small as 16 bytes per image) which have good object-recognition performance. In contrast to previous work in the domain of object recognition, we do not choose an arbitrary intermediate representation, but explicitly learn short codes. In contrast to previous approaches to learn compact codes, we optimize explicitly for (an upper bound on) classification performance. Optimization directly for binary features is difficult and nonconvex, but we present an alternation scheme and convex upper bound which demonstrate excellent performance in practice. PICODES of 256 bytes match the accuracy of the current best known classifier for the Caltech256 benchmark, but they decrease the database storage size by a factor of 100 and speed-up the training and testing of novel classes by orders of magnitude.

## 1 Introduction

In this work we consider the problem of efficient object-class recognition in large image collections. We are specifically interested in scenarios where the classes to be recognized are not known in advance. The motivating application is "object-class search by example" where a user provides at query time a small set of training images defining an arbitrary *novel* category and the system must retrieve from a large database images belonging to this class. This application scenario poses challenging requirements on the system design: the object classifier must be learned efficiently at query time from few examples; recognition must have low computational cost with respect to the database size; finally, compact image descriptors must be used to allow storage of large collections in memory rather than on disk for additional efficiency.

Traditional object categorization methods do not meet these requirements as they typically use non-linear kernels on high-dimensional descriptors, which renders them computationally expensive to train and test, and causes them to occupy large amounts of storage. For example, the LP-$\beta$ multiple kernel combiner [11] achieves state-of-the-art accuracy on several categorization benchmarks but it requires over 23 Kbytes to represent each image and it uses 39 feature-specific nonlinear kernels. This recognition model is impractical for our application because it would require costly *query-time* kernel evaluations for each image in the database since the training set varies with every new query and thus pre-calculation of kernel distances is not possible.

We propose to address these storage and efficiency requirements by learning a compact binary image representation, called PICODES[1], optimized to yield good categorization accuracy with linear

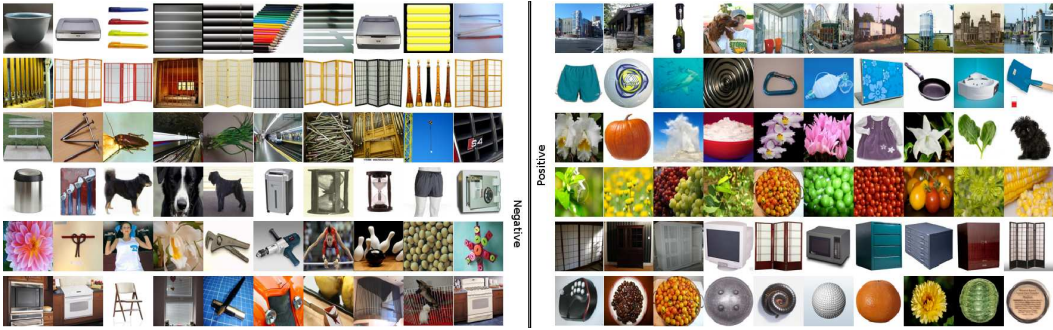

Figure 1: **Visualization of PICODES.** The 128-bit PICODE (whose accuracy on Caltech256 is displayed in figure 3) is applied to the test data of ILSVRC2010. Six of the 128 bits are illustrated as follows: for bit $c$, all images are sorted by non-binarized classifier outputs $a_c^\top x$ and the 10 smallest and largest are presented on each row. Note that $a_c$ is defined only up to sign, so the patterns to which the bits are specialized may appear in either the "positive" or "negative" columns.

(i.e., efficient) classifiers. The binary entries in our image descriptor are thresholded nonlinear projections of low-level visual features extracted from the image, such as descriptors encoding texture or the appearance of local image patches. Each non-linear projection can be viewed as implementing a nonlinear classifier using multiple kernels. The intuition is that we can then use these pre-learned multiple kernel combiners as a **classification basis** to define recognition models for arbitrary novel categories: the final classifier for a novel class is obtained by linearly combining the binary outputs of the basis classifiers, which we can pre-compute for every image in the database, thus enabling efficient novel object-class recognition even in large datasets.

The search for compact codes for images has been the subject of much recent work, which we loosely divide into "designed" and "learned" codes. In the former category we include min-hash [6], VLAD [14], and attributes [10, 18, 17] which are fully-supervised classifiers trained to recognize certain visual properties in the image. A related idea is the representation of images in terms of distances to basis classes. This has been previously investigated as a way to define image similarities [30], to perform video search [12], or to enable natural scene recognition and retrieval [29]. Torresani et al. [27] define a compact image code as a bitvector, the entries of which are the outputs of a large set of weakly-trained basis classifiers ("classemes") evaluated on the image. Simple linear classifiers trained on classeme vectors produce near state-of-the-art categorization accuracy. Li et al. [19] use the localized outputs of object detectors as an image representation. The advantage of this representation is that it encodes spatial information; furthermore, object detectors are more robust to clutter and uninformative background than classifiers evaluated on the entire image. These prior methods work under the assumption that an "overcomplete" representation for classification can be obtained by pre-learning classifiers for a large number of basis classes, some of which will be related to those encountered at test-time. Such high-dimensional representations are then compressed down using quantization, dimensionality reduction or feature selection methods.

The second strand of related work is the *learning* of compact codes for images [31, 26, 24, 15, 22, 8] where binary image codes are learned such that the Hamming distance between codewords approximates a kernelized distance between image descriptors, most typically GIST. Autoencoder learning [23], on the other hand, produces a compact code which has good image reconstruction properties, but again is not specialized for category recognition.

All the above descriptors can produce very compact codes, but few (excepting [27, 19]) have been shown to be effective at category-level recognition beyond simplified problems such as Caltech-20 [2] or Caltech-101 [14, 16]. In contrast, we consider Caltech-256 a baseline competence, and also test compact codes on a large-scale class retrieval task using ImageNet [7].

The goal of this paper then is to learn a compact binary code (as short as 128 bits) which has good object-category recognition accuracy. In contrast to previous learning approaches, our training objective is a direct approximation to this goal; while in contrast to previous "designed" descriptors, we learn abstract categories (see figure 1) aimed at optimizing classification rather than an arbitrary predefined set of attributes or classemes, and thus achieve increased accuracy for a given code length.

## 2   Technical approach

We start by introducing the basic classifier archi-
tecture used by state-of-the-art category recognizers,
which we want to leverage as effectively as possible
to define our image descriptor. Given an image $I$, a
bank of feature descriptors is computed (e.g. SIFT,
PHOG, GIST), to yield a feature vector $\boldsymbol{f}_I \in \mathbb{R}^F$
(the feature vector used in our implementation has
dimensionality $F = 17360$ and is described in the
experimental section).   State-of-the-art recognizers
use kernel matching between these descriptors to de-
fine powerful classifiers, nonlinear in $\boldsymbol{f}_I$. For exam-
ple, the LP-$\beta$ classifier of Gehler and Nowozin [11],
which has achieved the best results on several bench-
marks to date, operates by combining the outputs of
nonlinear SVMs trained on individual features. In
our work we approximate these nonlinear classifiers
by employing the lifting method of Vedaldi and Zis-
serman [28]: for the family of homogeneous additive
kernels $K$, there exists a finite-dimensional feature
map $\psi : \mathbb{R}^F \longrightarrow \mathbb{R}^{F(2r+1)}$ such that the nonlin-
ear kernel distance $K(\boldsymbol{f}_I, \boldsymbol{f}_{I'}) \approx \langle \psi(\boldsymbol{f}_I), \psi(\boldsymbol{f}_{I'}) \rangle$
where $r$ is a small positive integer (in our implemen-
tation set to 1).   These explicit feature maps allow
us to approximate a non-linear classifier, such as the

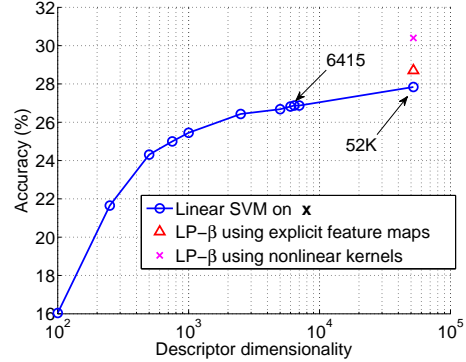

Figure 2: The accuracy versus compactness trade
off.  The benchmark is Caltech256, using 10 ex-
amples per class.  The pink cross shows the multi-
class categorization accuracy achieved by an LP-$\beta$
classifier using kernel distance; the red triangle is
the accuracy of an LP-$\beta$ classifier that uses "lifted-
up" features to approximate kernel distances; the
blue line shows accuracy of a linear SVM trained
on PCA projections of the lifted-up features, as a
function of PCA dimension.

LP-$\beta$ kernel combiner, via an efficient linear projection. As described below, we use these nonlinear
classifier approximated via linear projections as the basis for learning our features. However, in our
case $F(2r + 1) = 17360 \times 3 = 52080$. This dimensionality is too large in practice for our learn-
ing. Thus, we apply a linear dimensionality reduction $\boldsymbol{x}_I = \mathbf{P}\psi(\boldsymbol{f}_I)$, where projection matrix $\mathbf{P}$ is
obtained through PCA, so $\boldsymbol{x}_I \in \mathbb{R}^n$ for $n \ll F(2r + 1)$. As this procedure is performed identically
for every image we consider, we will drop the dependence on $I$ and simply refer to the "image" $\boldsymbol{x}$.

A natural question to address is: how much accuracy do we lose due to the kernel approximation
and the PCA projection? We answer this question in figure 2, where we compare the multi-class
classification accuracies obtained on the Caltech256 data set by the following methods using our
low-level descriptors $\boldsymbol{f} \in \mathbb{R}^{17360}$: an LP-$\beta$ combiner based on exact non-linear kernel calculations;
an LP-$\beta$ combiner using explicit feature maps; a linear SVM trained on the PCA projections $\boldsymbol{x}$ as a
function of the PCA subspace dimensionality. We see from this figure that the explicit maps degrade
the accuracy only slightly, which is consistent with the results reported in [28]. However, the linear
SVM produces slightly inferior accuracy even when applied to the full 52,080-dimensional feature
vectors. The key-difference between the linear SVM and the LP-$\beta$ classifier is that the former defines
a classifier in the joint space of all 13 features, while the latter first trains a separate classifier for
each feature, and then learns a linear combination of them. The results in our figure suggest that the
two-step procedure of LP-$\beta$ provides a form of beneficial regularization, a fact first noted in [11].
For our feature learning algorithm, we chose to use a PCA subspace of dimensionality $n = 6415$
since, as suggested by the plot, this setting gives a good tradeoff in terms of compact dimensionality
and good recognition accuracy.

Torresani et al. [27] have shown that an effective image descriptor for categorization can be built
by collecting in a vector the thresholded outputs of a large set of nonlinear classifiers evaluated on
the image. This "classeme" descriptor can produce recognition accuracies within 10% of the state
of the art for novel classes even with simple linear classification models. Using our formulation
based on explicit feature maps, we can approximately express each classeme entry (which in  [27]
is implemented as an LP-$\beta$ classifier) as the output of a linear classifier

$$h(\boldsymbol{x}; \boldsymbol{a}_c) = \mathbf{1}[\boldsymbol{a}_c^T \boldsymbol{x} > 0] \tag{1}$$

where $\mathbf{1}[.]$ is the 0-1 indicator function of its boolean argument and $\boldsymbol{x}$ is the PCA-projection of $\psi(\boldsymbol{f})$,
with a 1 appended to it to avoid dealing with an explicit bias coefficient, i.e., $\boldsymbol{x} = [\mathbf{P}\psi(\boldsymbol{f}); 1]$ .

If following the approach of Torresani et al. [27], we would collect $C$ training categories, and learn the parameters $\boldsymbol{a}_c$ for each class from offline training data using some standard training objective such as hinge loss. We gather the parameters into a $n \times C$ matrix

$$\boldsymbol{A} = [\boldsymbol{a}_1 | \dots | \boldsymbol{a}_C].$$

Then, for image $\boldsymbol{x}$, the "classeme" descriptor $\boldsymbol{h}(\boldsymbol{x})$ is computed as the concatenation of the outputs of the classifiers learned for the training categories:

$$\boldsymbol{h}(\boldsymbol{x}; \boldsymbol{A}) = \begin{bmatrix} h(\boldsymbol{x}; \boldsymbol{a}_1) \\ \vdots \\ h(\boldsymbol{x}; \boldsymbol{a}_C) \end{bmatrix} \in \{0, 1\}^C \tag{2}$$

The PICODE descriptor is also of this form. However, the key-difference with respect to [27] lies in our training procedure, and the fact that the dimensionality $C$ is no longer restricted to be the same as the number of training classes.

To emphasize once more the contributions of this paper, let us review the shortcomings of existing attribute- and classifier-based descriptors, which we overcome in this paper:

- Prior work used attributes learned disjointly from one another, which "just so-happen" to work well as features for classification, without theoretical justification for their use in subsequent classification. Given that we want to use attributes as features for linear classification, we propose to formalize as learning objective that linear combinations of such attributes must yield good accuracy.

- Unlike the attribute or classeme approach, our method decouples the number of training classes from the target dimensionality of the binary descriptor. We can optimize our features for any arbitrary desired length, thus avoiding a suboptimal feature selection stage.

- Finally, we directly optimize the learning parameters with respect to *binary* features while prior attribute systems binarized the features in a quantization stage after the learning.

We now introduce a framework to learn the $\boldsymbol{A}$ parameters directly on a classification objective.

## 2.1 Learning the basis classifiers

We assume that we are given a set of $N$ training images, with each image coming from one of $K$ training classes. We will continue to let $C$ stand for the dimensionality (i.e., number of bits) of our code. Let $\mathcal{D} = \{(\boldsymbol{x}_i, \boldsymbol{y}_i)\}_{i=1}^N$ be the training set for learning the basis classifiers, where $\boldsymbol{x}_i$ is the $i$-th image example (represented by its $n$-dimensional PCA projection augmented with a constant entry set to 1) and $\boldsymbol{y}_i \in \{-1, +1\}^K$ is a vector encoding the category label out of $K$ possible classes: $y_{ik} = +1$ iff the $i$-th example belongs to class $k$.

We then define our $c$-th basis classifier to be a boolean function of the form (1), a thresholded *nonlinear* projection of the original low-level features $\boldsymbol{f}$, parameterized by $\boldsymbol{a}_c \in \mathbf{R}^n$. We then optimize these parameters so that linear combinations of these basis classifiers yield good categorization accuracy on $\mathcal{D}$. The learning objective introduces auxiliary variables $(\boldsymbol{w}_k, b_k)$ for each training class, which parameterize the linear classifier for that training class, operating on the PICODE representation of the training examples, and the objective for $\boldsymbol{A}$ simply minimizes over these auxiliaries:

$$E(\boldsymbol{A}) = \min_{\boldsymbol{w}_{1..K}, b_{1..K}} E(\boldsymbol{A}, \boldsymbol{w}_{1..K}, b_{1..K}) \tag{3}$$

Solving for $\boldsymbol{A}$ then amounts to simultaneous optimization over all variables of the following learning objective, which is a trade off between a small classification error and a large margin when using the output bits of the basis classifiers as features in a one-versus-all linear SVM:

$$E(\boldsymbol{A}, \boldsymbol{w}_{1..K}, b_{1..K}) = \sum_{k=1}^K \left\{ \frac{1}{2} \|\boldsymbol{w}_k\|^2 + \frac{\lambda}{N} \sum_{i=1}^N \ell \left[ y_{i,k}(b_k + \boldsymbol{w}_k^\top \boldsymbol{h}(\boldsymbol{x}_i; \boldsymbol{A})] \right] \right\} \tag{4}$$

where $\ell[\cdot]$ is the traditional hinge loss function. Expanding, we get

$$E(\boldsymbol{A}, \boldsymbol{w}_{1..K}, b_{1..K}) = \sum_{k=1}^K \left\{ \frac{1}{2} \|\boldsymbol{w}_k\|^2 + \frac{\lambda}{N} \sum_{i=1}^N \ell \left[ y_{i,k}(b_k + \sum_{c=1}^C w_{kc} \mathbf{1}[\boldsymbol{a}_c^T \boldsymbol{x}_i > 0]) \right] \right\} .$$

Note that the linear SVM and the basis classifiers are learned jointly using the method described below.

## 2.2 Optimization

We propose to minimize this error function by block coordinate descent. We alternate between the two following steps:

**1. Learn classifiers.**
We fix $\boldsymbol{A}$ and optimize the objective with respect to $\boldsymbol{w}$ and $\boldsymbol{b}$ jointly. This optimization is convex and equivalent to traditional linear SVM learning.

**2. Learn projectors.**
Given the current values of $\boldsymbol{w}$ and $\boldsymbol{b}$, we minimize the objective with respect to $\boldsymbol{A}$ by updating one basis-classifier at a time. Let us consider the update of $\boldsymbol{a}_c$ with fixed parameters $\boldsymbol{w}_{1..K}, \boldsymbol{b}, \boldsymbol{a}_1, \ldots, \boldsymbol{a}_{c-1}, \boldsymbol{a}_{c+1}, \ldots, \boldsymbol{a}_C$. It can be seen (Appendix A) that in this case the objective becomes:

$$E(\boldsymbol{a}_c) = \sum_{i=1}^{N} v_i \mathbf{1}[z_i \boldsymbol{a}_c^T \boldsymbol{x}_i > 0] + const \tag{5}$$

where $z_i \in \{-1, +1\}$ and $v_i \in \mathbb{R}^+$ are known values computed from the fixed parameters. Optimizing the objective in Eq. 5 is equivalent to learning a linear classifier minimizing the sum of weighted misclassifications, where $v_i$ represents the cost of misclassifying example $i$. Unfortunately, this objective is not convex and it is difficult to optimize. Thus, we replace it with the following convex upper bound defined in terms of the hinge function $\ell$:

$$\hat{E}(\boldsymbol{a}_c) = \sum_{i=1}^{N} v_i \ell(z_i \boldsymbol{a}_c^T \boldsymbol{x}_i) \tag{6}$$

This objective can be globally optimized using an LP solver or software for SVM training. We had success with LIBLINEAR [9], which deals nicely with the large problem sizes we considered.

We have also experimented with several other optimization methods, including stochastic gradient descent applied to a modified version of our objective where we replaced the binarization function $h(\boldsymbol{x}; \boldsymbol{a}_c) = \mathbf{1}[\boldsymbol{a}_c^T \boldsymbol{x} > 0]$ with the sigmoid function $\sigma(\boldsymbol{x}; \boldsymbol{a}_c) = 1/(1 + \exp(-\frac{2}{T} \boldsymbol{a}_c^T \boldsymbol{x}))$ to relax the problem. After learning, at test-time we replaced back $\sigma(\boldsymbol{x}; \boldsymbol{a}_c)$ with $h(\boldsymbol{x}; \boldsymbol{a}_c)$ to obtain binary descriptors. However, we found that these binary codes performed much worse than those directly learned via the coordinate descent procedure described above.

## 3 Experiments

We now describe experimental evaluations carried out over several data sets. In order to allow a fair comparison, we reimplemented the "classeme descriptor" based on the same set of low-level features and settings described in [27] but using the explicit feature map framework to replace the expensive nonlinear kernel distance computations. The low-level features are: color GIST [21], spatial pyramid of histograms of oriented gradients (PHOG) [4], spatial pyramid of self-similarity descriptors [25], and a histogram of SIFT features [20] quantized using a dictionary of 5000 visual words. Each spatial pyramid level of each descriptor was treated as a separate feature, thus producing a total of 13 low-level features. Each of these features was lifted up to a higher-dimensional space using the explicit feature maps of Vedaldi and Zisserman [28]. We chose the mapping approximating the histogram intersection kernels for $n = 1$, which effectively mapped each low-level feature descriptor to a space 3 times larger than its original one. The resulting vectors $\psi(\boldsymbol{f})$ have dimensionality $3 \times F = 52,080$. To learn our basis classifiers, we used 6415-dimensional PCA projections of these high-dimensional vectors.

We compared PICODES with binary classeme vectors. For both descriptors we used a training set of $K = 2659$ classes randomly sampled from the ImageNet dataset [7], with 30 images for each category for a total of $N = 2659 \times 30 = 79,770$ images. Each class in ImageNet is associated to a "synset", which is a set of words describing the category. Since we wanted to evaluate the

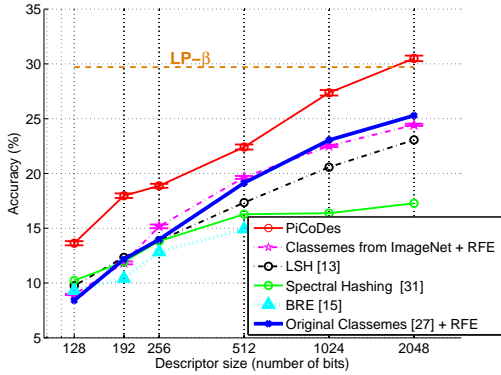

Figure 3: Multiclass categorization accuracy on Caltech256 using different binary codes, as a function of the number of bits. PICODES outperform all the other compact codes. PICODES of 2048 bits match the accuracy of the state-of-the-art LP-$\beta$ classifier.

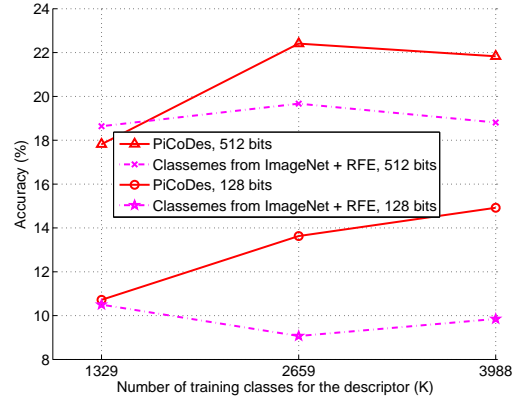

Figure 4: Caltech256 classification accuracy for PICODES and classemes as a function of the number of training classes used to learn the descriptors.

learned descriptors on the Caltech256 and ILSVRC2010 [3] benchmarks, we selected 2659 ImageNet training classes such that the synsets of these classes do not contain any of the Caltech256 or ILSVRC2010 class labels, so as to avoid "pre-learning" the test classes during the feature-training stage, which could yield a biased evaluation.

We also present comparisons with binary codes trained to directly approximate the Eucliden distances between the vectors $x$, using the following previously proposed algorithms: locality sensitive hashing (LSH) [13], spectral hashing (SH) [31], and binary reconstructive embeddings (BRE) [15]. Since these descriptors in the past have been used predominantly with the k-NN classifier, we have also tested this classification model but obtained inferior results compared to when using a linear SVM. For this reason, here we report only results using the linear SVM model.

**Multiclass recognition using PICODES.** We first report in figure 3 the results showing multiclass classification accuracy achieved with binary codes on the Caltech256 data set. Since PICODES are optimized for categorization using linear models, we adopt simple linear "one-versus-all" SVMs as classifiers. For each Caltech256 category, the classifier was trained using 10 positive examples and a total of 2550 negative examples obtained by sampling 10 images from each of the other classes. We computed accuracies using 25 test examples per class, using 5-fold cross validation for the model selection. As usual, accuracy is computed as the average over the mean recognition rates per class. Figure 3 shows the results obtained with binary descriptors of varying dimensionality. While our approach can accommodate easily the case were the number of feature dimensions ($C$) is different from the number of feature-training categories ($K$), the classeme learning method can only produce descriptors of size $K$. Thus, the descriptor size is typically reduced through a subsequent feature selection stage [27, 19]. In this figure we show accuracy obtained with classeme features selected by multi-class recursive feature elimination (RFE) with SVM [5], which at each iteration retrains the SVMs for all classes on the active features and then removes the $m$ least-used active features until reaching the desired compactness. We also report accuracy obtained with the original classeme vectors of [27], which were learned with exact kernels on a different training set, consisting of weakly-supervised images retrieved with text-based image search engines. From this figure we see that PICODES greatly outperform all the other compact codes considered here (classemes, LSH, SH, BRE) for all descriptor sizes. In addition, perhaps surprisingly, PICODES of 2048 bits yield even higher accuracy than the-state-of-the-art multiple kernel combiner LP-$\beta$ [11] trained on our low-level features $f$ (30.5% versus 29.7%). At the same time, our codes are 100 times smaller and reduce the training and testing time by two-orders of magnitude compared to LP-$\beta$.

We have also investigated the influence of the parameter $K$, i.e., the number of training classes used to learn the descriptor. We learned different PICODES and classeme descriptors by varying $K$ while keeping the number of training examples per class fixed to 30. Figure 4 shows the multiclass categorization accuracy on Caltech256 as a function of $K$. From this plot we see that PICODES

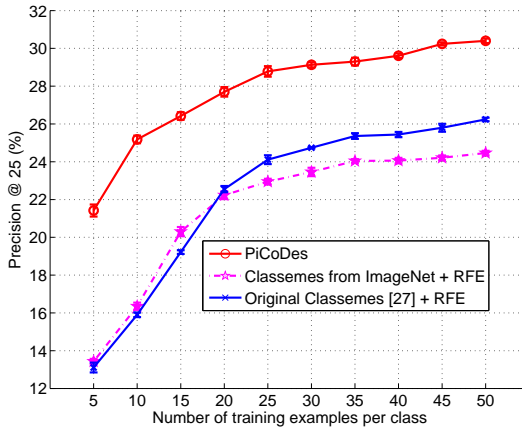
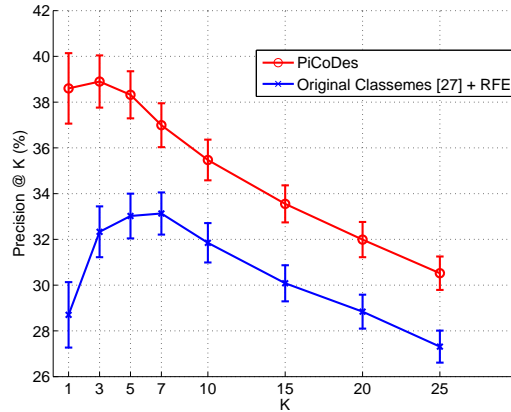

Figure 5: Precision of object-class search using codes of 256 bytes on Caltech256: for a varying number of training examples per class, we report the percentage of true positives in top 25 retrieved from a dataset containing 6375 distractors and 25 relevant results.

Figure 6: Finding pictures of an object class in the ILSVRC2010 dataset, which includes 150K images for 1000 different classes, using 256-byte codes. PICODES enable accurate class retrieval from this large collection in less than a second.

profit more than classemes from a larger number of training classes, producing further improvement in generalization on novel classes.

An advantage of the classeme learning setup presented in [27] is the intrinsic parallelization that can be achieved during the learning of the $C$ classeme classifiers (which are disjointly trained), enabling the use of more training data. We have considered this scenario, and tried learning the classeme descriptors from ImageNet using *5 times* more images than for PICODES, i.e., 150 images for each training category for a total of $N = 2659 \times 150 = 398,850$ examples. Despite the disparate training set sizes, we found that PICODES still outperformed classemes (22.41% versus 20.4% for 512 bits).

**Retrieval of object classes on Caltech256.** In figure 5 we present results corresponding to our motivating application of object-class search, using codes of 256 bytes. For each Caltech256 class, we trained a one-versus-all linear SVM using $p$ positive examples and $p \times 255$ negative examples, for varying values of $p$. We then used the learned classifier to find images of that category in a database containing 6400 Caltech256 test images, with 25 images per class. The retrieval accuracy is measured as precision at 25, which is the proportion of true positives (i.e., images of the query class) ranked in the top 25. Again, we see that our features yield consistently better ranking precision compared to classeme vectors learned on the same ImageNet training set, and produce an average improvement of about 28% over the original classeme features.

**Object class search in a large image collection.** Finally, we present experiments on the 150K image data set of the Large Scale Visual Recognition Challenge 2010 (ILSVRC2010) [3], which includes images of 1000 different categories, different from those used to train PICODES. Again, we evaluate our binary codes on the task of object-class retrieval. For each of the 1000 classes, we train a linear SVM using all examples of that class available in the ILSVRC2010 training set (this number varies from a minimum of 619 to a maximum of 3047, depending on the class) and 4995 negative examples obtained by sampling five images from each of the other classes. We test the classifiers on the ILSVRC2010 test set, which includes 150 images for each of the 1000 classes. Figure 6 shows a comparison between PICODES and classemes in terms of precision at $k$ for varying $k$. Despite the very large number of distractors (149,850 for each query), search with our codes yields precisions exceeding 38%. Furthermore, the tiny size of our descriptor allows the entire data set to be easily kept in memory for efficient retrieval (the whole database size using our representation is only 36MB): the average search time for a query class, including the learning time, is about 1 second on an Intel Xeon X5680 @ 3.33GHz.

## 4 Conclusion

We have described a new type of compact code, which is learned by directly minimizing a multiclass classification objective on a large set of offline training classes. This allows recognition of novel categories to be performed using extremely compact codes with state-of-the-art accuracy. Although there is much existing work on learning compact codes, we know of no other compact code which offers this performance on a category recognition task.

Our experiments have focussed on whole-image "Caltech-like" category recognition, while it is clear that subimage recognition is also an important application. However, we argue that for many image search tasks, whole-image performance is relevant, and for a very compact code, one could possibly encode several windows (dozens, say) in each image, while retaining a relatively compact representation.

Additional material including software to extract PICODES from images may be obtained from [1].

**Acknowledgments**

We are grateful to Chen Fang for programming help. This research was funded in part by Microsoft and NSF CAREER award IIS-0952943.

## A Derivation of eq. 5

We present below the derivation of eq. 5. First, we rewrite our objective function, i.e., eq. 4, in expanded form:

$$E(\boldsymbol{A}, \boldsymbol{w}_{1..K}, b_{1..K}) = \sum_{k=1}^{K} \left\{ \frac{1}{2} \|\boldsymbol{w}_k\|^2 + \frac{\lambda}{N} \sum_{i=1}^{N} \ell \left[ y_{ik}(b_k + \sum_{c=1}^{C} w_{kc} \mathbf{1}[\boldsymbol{a}_c^T \boldsymbol{x}_i > 0]) \right] \right\}.$$

Fixing the parameters $\boldsymbol{w}_{1..K}, \boldsymbol{b}, \boldsymbol{a}_1, \ldots, \boldsymbol{a}_{c-1}, \boldsymbol{a}_{c+1}, \ldots, \boldsymbol{a}_C$ and minimizing the function above with respect to $\boldsymbol{a}_c$, is equivalent to minimizing the following objective:

$$E'(\boldsymbol{a}_c) = \sum_{k=1}^{K} \sum_{i=1}^{N} \ell \left[ y_{ik} w_{kc} \mathbf{1}[\boldsymbol{a}_c^T \boldsymbol{x}_i > 0] + y_{ik} b_k + \sum_{c' \neq c} y_{ik} w_{kc'} \mathbf{1}[\boldsymbol{a}_{c'}^T \boldsymbol{x}_i > 0] \right].$$

Let us define $\alpha_{ikc} \equiv y_{ik} w_{kc}$, and $\beta_{ikc} \equiv (y_{ik} b_k + \sum_{c' \neq c} y_{ik} w_{kc'} \mathbf{1}[\boldsymbol{a}_{c'}^T \boldsymbol{x}_i > 0])$. Then, we can rewrite the objective as follows:

$$
\begin{aligned}
E'(\boldsymbol{a}_c) &= \sum_{k=1}^{K} \sum_{i=1}^{N} \ell \left[ \alpha_{ikc} \mathbf{1}[\boldsymbol{a}_c^T \boldsymbol{x}_i > 0] + \beta_{ikc} \right] \\
&= \sum_{i=1}^{N} \left\{ \mathbf{1}[\boldsymbol{a}_c^T \boldsymbol{x}_i > 0] \sum_{k=1}^{K} \ell(\alpha_{ikc} + \beta_{ikc}) + (1 - \mathbf{1}[\boldsymbol{a}_c^T \boldsymbol{x}_i > 0]) \sum_{k=1}^{K} \ell(\beta_{ikc}) \right\} \\
&= \sum_{i=1}^{N} \left\{ \mathbf{1}[\boldsymbol{a}_c^T \boldsymbol{x}_i > 0] \sum_{k=1}^{K} \ell(\alpha_{ikc} + \beta_{ikc}) - \ell(\beta_{ikc}) \right\} + const .
\end{aligned}
$$

Finally, it can be seen that optimizing this objective is equivalent to minimizing

$$E(\boldsymbol{a}_c) = \sum_{i=1}^{N} v_i \mathbf{1}[z_i \boldsymbol{a}_c^T \boldsymbol{x}_i > 0]$$

where $v_i = \left| \sum_{k=1}^{K} \ell(\alpha_{ikc} + \beta_{ikc}) - \ell(\beta_{ikc}) \right|$ and $z_i = \text{sign} \left( \sum_{k=1}^{K} \ell(\alpha_{ikc} + \beta_{ikc}) - \ell(\beta_{ikc}) \right)$. This yields eq. 5.

## Footnotes

[1]Which we think of as "Picture Codes" or "Pico-Descriptors", or (with Greek pronunciation) $\pi$-codes

# References

[1] http://vlg.cs.dartmouth.edu/picodes.

[2] B. Babenko, S. Branson, and S. Belongie. Similarity metrics for categorization: From monolithic to category specific. In *Intl. Conf. Computer Vision*, pages 293 –300, 2009.

[3] A. Berg, J. Deng, and L. Fei-Fei. Large scale visual recognition challenge, 2010. http://www.image-net.org/challenges/LSVRC/2010/.

[4] A. Bosch, A. Zisserman, and X. Muñoz. Representing shape with a spatial pyramid kernel. In *Conf. Image and Video Retrieval (CIVR)*, pages 401–408, 2007.

[5] O. Chapelle and S. S. Keerthi. Multi-class feature selection with support vector machines. *Proc. of the Am. Stat. Assoc.*, 2008.

[6] O. Chum, J. Philbin, and A. Zisserman. Near duplicate image detection: min-hash and tf-idf weighting. In *British Machine Vision Conf.*, 2008.

[7] J. Deng, W. Dong, R. Socher, L.-J. Li, K. Li, and L. Fei-Fei. ImageNet: A Large-Scale Hierarchical Image Database. In *CVPR*, 2009.

[8] M. Douze, A. Ramisa, and C. Schmid. Combining attributes and fisher vectors for efficient image retrieval. In *Proc. Comp. Vision Pattern Recogn. (CVPR)*, 2011.

[9] R.-E. Fan, K.-W. Chang, C.-J. Hsieh, X.-R. Wang, and C.-J. Lin. Liblinear: A library for large linear classification. *Journal of Machine Learning Research*, 9:1871–1874, 2008.

[10] A. Farhadi, I. Endres, D. Hoiem, and D. Forsyth. Describing objects by their attributes. In *CVPR*, 2009.

[11] P. Gehler and S. Nowozin. On feature combination for multiclass object classification. In *ICCV*, 2009.

[12] A. G. Hauptmann, R. Yan, W.-H. Lin, M. G. Christel, and H. D. Wactlar. Can high-level concepts fill the semantic gap in video retrieval? a case study with broadcast news. *IEEE Transactions on Multimedia*, 9(5):958–966, 2007.

[13] P. Indyk and R. Motwani. Approximate nearest neighbors: towards removing the curse of dimensionality. In *STOC '98: Proceedings of the thirtieth annual ACM symposium on Theory of computing*, New York, NY, USA, 1998. ACM Press.

[14] H. Jégou, M. Douze, C. Schmid, and P. Pérez. Aggregating local descriptors into a compact image representation. In *Proc. Comp. Vision Pattern Recogn. (CVPR)*, 2010.

[15] B. Kulis and T. Darrell. Learning to hash with binary reconstructive embeddings. In *Advances in Neural Information Processing Systems (NIPS)*, 2009.

[16] B. Kulis and K. Grauman. Kernelized locality-sensitive hashing for scalable image search. In *Intl. Conf. Computer Vision*, 2010.

[17] N. Kumar, A. C. Berg, P. N. Belhumeur, and S. K. Nayar. Attribute and Simile Classifiers for Face Verification. In *Intl. Conf. Computer Vision*, 2009.

[18] C. H. Lampert, H. Nickisch, and S. Harmeling. Learning to detect unseen object classes by between-class attribute transfer. In *CVPR*, 2009.

[19] L. Li, H. Su, E. Xing, and L. Fei-Fei. Object Bank: A high-level image representation for scene classification & semantic feature sparsification. In *NIPS*. 2010.

[20] D. Lowe. Distinctive image features from scale-invariant keypoints. *Intl. Jrnl. of Computer Vision*, 60(2):91–110, 2004.

[21] A. Oliva and A. Torralba. Building the gist of a scene: The role of global image features in recognition. *Visual Perception, Progress in Brain Research*, 155, 2006.

[22] M. Raginsky and S. Lazebnik. Locality-sensitive binary codes from shift-invariant kernels. In *Advances in Neural Information Processing Systems (NIPS)*, 2010.

[23] M. Ranzato, Y. Boureau, and Y. LeCun. Sparse feature learning for deep belief networks. In *Advances in Neural Information Processing Systems (NIPS)*, 2007.

[24] R. Salakhutdinov and G. Hinton. Semantic hashing. *Int. J. Approx. Reasoning*, 50:969–978, 2009.

[25] E. Shechtman and M. Irani. Matching local self-similarities across images and videos. In *Proc. Comp. Vision Pattern Recogn. (CVPR)*, 2007.

[26] A. Torralba, R. Fergus, and Y. Weiss. Small codes and large image databases for recognition. In *Proc. Comp. Vision Pattern Recogn. (CVPR)*, 2008.

[27] L. Torresani, M. Szummer, and A. Fitzgibbon. Efficient object category recognition using classemes. In *ECCV*, 2010.

[28] A. Vedaldi and A. Zisserman. Efficient additive kernels via explicit feature maps. In *CVPR*, 2010.

[29] J. Vogel and B. Schiele. Semantic modeling of natural scenes for content-based image retrieval. *Intl. Jrnl. of Computer Vision*, 72(2):133–157, 2007.

[30] G. Wang, D. Hoiem, and D. Forsyth. Learning image similarity from flickr using stochastic intersection kernel machines. In *Intl. Conf. Computer Vision*, 2009.

[31] Y. Weiss, A. Torralba, and R. Fergus. Spectral hashing. In *NIPS*. 2009.

